# Sparse Coding of Natural Images Using an Overcomplete Set of Limited Capacity Units

**Eizaburo Doi**
Center for the Neural Basis of Cognition
Carnegie Mellon University
Pittsburgh, PA 15213
edoi@cnbc.cmu.edu

**Michael S. Lewicki**
Center for the Neural Basis of Cognition
Computer Science Department
Carnegie Mellon University
Pittsburgh, PA 15213
lewicki@cnbc.cmu.edu

## Abstract

It has been suggested that the primary goal of the sensory system is to represent input in such a way as to reduce the high degree of redundancy. Given a noisy neural representation, however, solely reducing redundancy is not desirable, since redundancy is the only clue to reduce the effects of noise. Here we propose a model that best balances redundancy reduction and redundant representation. Like previous models, our model accounts for the localized and oriented structure of simple cells, but it also predicts a different organization for the population. With noisy, limited-capacity units, the optimal representation becomes an overcomplete, multi-scale representation, which, compared to previous models, is in closer agreement with physiological data. These results offer a new perspective on the expansion of the number of neurons from retina to V1 and provide a theoretical model of incorporating useful redundancy into efficient neural representations.

## 1 Introduction

Efficient coding theory posits that one of the primary goals of sensory coding is to eliminate redundancy from raw sensory signals, ideally representing the input by a set of statistically independent features [1]. Models for learning efficient codes, such as sparse coding [2] or ICA [3], predict the localized, oriented, and band-pass characteristics of simple cells. In this framework, units are assumed to be non-redundant and so the number of units should be identical to the dimensionality of the data.

Redundancy, however, can be beneficial if it is used to compensate for inherent noise in the system [4]. The models above assume that the system noise is low and negligible so that redundancy in the representation is not necessary. This is equivalent to assuming that the representational capacity of individual units is unlimited. Real neurons, however, have limited capacity [5], and this should place constraints on how a neural population can best encode a sensory signal. In fact, there are important characteristics of simple cells, such as the multi-scale representation, that cannot be explained by efficient coding theory.

The aim of this study is to evaluate how the optimal representation changes when the system

is constrained by limited capacity units. We propose a model that best balances redundancy reduction and redundant representation given the limited capacity units. In contrast to the efficient coding models, it is possible to have a larger number of units than the intrinsic dimensionality of the data. This further allows to introduce redundancy in the population, enabling precise reconstruction using the imprecise representation of a single unit.

## 2   Model

**Encoding**

We assume that the encoding is a linear transform of the input $\mathbf{x}$, followed by the additive *channel* noise $\mathbf{n} \sim \mathcal{N}(\mathbf{0},\, \sigma_n^2 \mathbf{I})$,

$$\mathbf{r} = \mathbf{Wx} + \mathbf{n} \tag{1}$$
$$= \mathbf{u} + \mathbf{n}, \tag{2}$$

where rows of $\mathbf{W}$ are referred to as the analysis vectors, $\mathbf{r}$ is the representation, and $\mathbf{u}$ is the signal component of the representation. We will refer to $\mathbf{u}$ as coefficients because it is a set of clean coefficients associated with the synthesis vectors in the decoding process, as described below.

We define channel noise level as follows,

$$(\text{channel noise level}) = \frac{\sigma_n^2}{\sigma_t^2} \times 100 \ [\%] \tag{3}$$

where $\sigma_t^2$ is a constant target value of the coefficient variance. It is the inverse of the signal-to-noise ratio in the representation, and therefore, we can control the information capacity of a single unit by varying the channel noise variance. Note that in the previous models [2, 3, 6] there is no channel noise; therefore $\mathbf{r} = \mathbf{u}$, where the signal-to-noise ratio of the representation is infinite.

**Decoding**

The decoding process is assumed to be a linear transform of the representation,

$$\hat{\mathbf{x}} = \mathbf{Ar}, \tag{4}$$

where the columns of $\mathbf{A}$ are referred to as the synthesis vectors[1], and $\hat{\mathbf{x}}$ is the reconstruction of the input. The reconstruction error $\mathbf{e}$ is then expressed as

$$\mathbf{e} = \mathbf{x} - \hat{\mathbf{x}} \tag{5}$$
$$= (\mathbf{I} - \mathbf{AW})\,\mathbf{x} - \mathbf{An}. \tag{6}$$

Note that no assumption on the reconstruction error is made, because eq. 4 is not a probabilistic data generative model, in contrast to the previous approaches [2, 6].

**Representation desiderata**

We assume a two-fold goal for the representation. The first is to preserve input information a given noisy, limited information capacity unit. The second is to make the representation

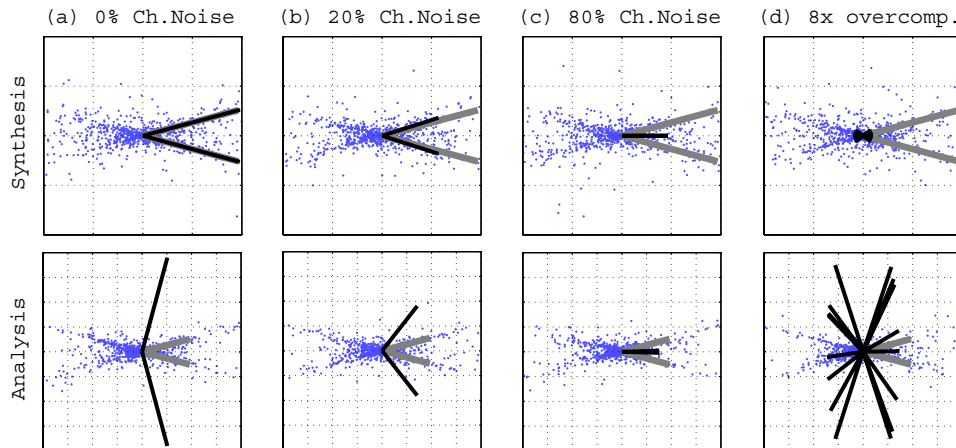

Figure 1: Optimal codes for toy problems. Data (shown with small dots) is generated with two i.i.d. Laplacians mixed via non-orthogonal basis functions (shown by gray bars). The optimal synthesis vectors (top row) and analysis vectors (bottom row) are shown as black bars. Plots of synthesis vectors are scaled for visibility. (a-c) shows the complete code with 0, 20, and 80% channel noise level. (d) shows the case of 80% channel noise using an 8x overcomplete code. Reconstruction error is (a) 0.0%, (b) 13.6%, (c) 32.2%, (d) 6.8%.

as sparse as possible, which yields an efficient code. The cost function to be minimized is therefore defined as follows,

$$C(\mathbf{A}, \mathbf{W}) = (\text{reconstruction error}) - \lambda_1(\text{sparseness}) + \lambda_2(\text{fixed variance}) \quad (7)$$

$$= \langle \|\mathbf{e}\|^2 \rangle - \lambda_1 \sum_{i=1}^{M} \langle \ln p(u_i) \rangle + \lambda_2 \sum_{i=1}^{M} \left\{ \ln \left[ \frac{\langle u_i^2 \rangle}{\sigma_t^2} \right] \right\}^2, \quad (8)$$

where $\langle \cdot \rangle$ represents an ensemble average over the samples, and $M$ is the number of units. The sparseness is measured by the loglikelihood of a sparse prior $p$ as in the previous models [2, 3, 6]. The third, fixed variance term penalizes the case in which the coefficient variance of the i-th unit $\langle u_i^2 \rangle$ deviates from its target value $\sigma_t^2$. It serves to fix the signal-to-noise ratio in the representation, yielding a fixed information capacity. Without this term, the coefficient variance could become trivially large so that the signal-to-noise ratio is high, yielding smaller reconstruction error; or, the variance becomes small to satisfy only the sparseness constraint, which is not desirable either.

Note that in order to introduce redundancy in the representation, we do not assume statistical independence of the coefficients. The second term in eq. 8 measures the sparseness of coefficients individually but it does not impose their statistical independence. We illustrate it with toy problems in Figure 1. If there is no channel noise, the optimal complete (1x) code is identical to the ICA solution (a), since it gives the most sparse, non-Gaussian solution with minimal error. As the channel noise increases (b and c), sparseness is compromised for minimizing the reconstruction error by choosing correlated, redundant representation. In an extreme case where the channel noise is high enough, the two units are almost completely redundant (c). It should be noted that in such a case two vectors represent the direction of the first principal component of the data.

In addition to de-emphasizing sparseness, there is another way to introduce redundancy in the representation. Since the goal of the representation is not the separation of independent sources, we can set an arbitrarily large number of units in the representation. When the information capacity of a single unit is limited, the capacity of a population can be made large

by increasing the number of units. As shown in Figure 1c-d, the reconstruction error decreases as we increase the degree of overcompleteness. Note that the optimal overcomplete code is not simply a duplication of the complete code.

**Learning rule**

The optimal code can be learned by the gradient descent of the cost function (eq. 8) with respect to $\mathbf{A}$ and $\mathbf{W}$,

$$\Delta \mathbf{A} \quad \propto \quad (\mathbf{I} - \mathbf{AW}) \left\langle \mathbf{xx}^T \right\rangle \mathbf{W}^T - \sigma_n^2 \mathbf{A}, \tag{9}$$

$$\Delta \mathbf{W} \quad \propto \quad \mathbf{A}^T(\mathbf{I} - \mathbf{AW}) \left\langle \mathbf{xx}^T \right\rangle$$

$$+ \lambda_1 \left\langle \frac{\partial \ln(\mathbf{u})}{\partial \mathbf{u}} \mathbf{x}^T \right\rangle - \lambda_2 \mathrm{diag} \left( \frac{\ln[\langle \mathbf{u}^2 \rangle / \sigma_t^2]}{\langle \mathbf{u}^2 \rangle} \right) \mathbf{W} \left\langle \mathbf{xx}^T \right\rangle. \tag{10}$$

In the limit of zero channel noise in the square case (e.g., Figure 1a) the solution is at the equilibrium when $\mathbf{W} = \mathbf{A}^{-1}$ (see eq. 9), where the learning rule becomes similar to the standard ICA (except the 3rd term in eq. 10). In all other cases, there is no reason to believe that $\mathbf{W} = \mathbf{A}^{-1}$, if it exists, minimizes the cost function. This is the reason why we need to optimize $\mathbf{A}$ and $\mathbf{W}$ individually.

# 3   Optimal representations for natural images

We examined optimal codes for natural image patches using the proposed model. The training data is 8x8 pixel image patches, sampled from a data set of 62 natural images [7]. The data is not preprocessed except for the subtraction of DC components [8]. Accordingly, the intrinsic dimensionality of the data is 63, and an N-times overcomplete code consists of N×63 units. The training set is sequentially updated during the learning and the order is randomized to prevent any local structure in the sequence. A typical number of image patches in a training is $5 \times 10^6$.

Here we first descirbe how the presence of channel noise changes the optimal code in the complete case. Next, we examine the optimal code at different degree of overcompleteness given a high channel noise level.

## 3.1   Optimal code at different channel noise level

We varied the channel noise level as 10, 20, 40, and 80%. As shown in Figure 2, learned synthesis and analysis vectors look somewhat similar to ICA (only 10 and 80% are shown for clarity). The comparison to the receptive fields of simple cells should be made with the analysis vectors [9, 10, 7]. They show localized and oriented structures and are well fitted by the Gabor function, indicating the similarity to simple cells in V1. Now, an additional characteristic to the Gabor-like structure is that the spatial-frequency tuning of the analysis vectors shifts towards lower spatial-frequencies as the channel noise increases (Figure 2d).

The learned code is expected to be robust to the channel noise. The reconstruction error with respect to the data variance turned out to be 6.5, 10.1, 15.7, and 23.8% for 10, 20, 40, and 80% of channel noise level, respectively. The noise reduction is significant considering the fact that any whitened representation including ICA should generate the reconstruction error of exactly the same amount of the channel noise level[2]. For the learned ICA code shown in Figure 2a, the reconstruction error was 82.7% when 80% channel noise was applied.

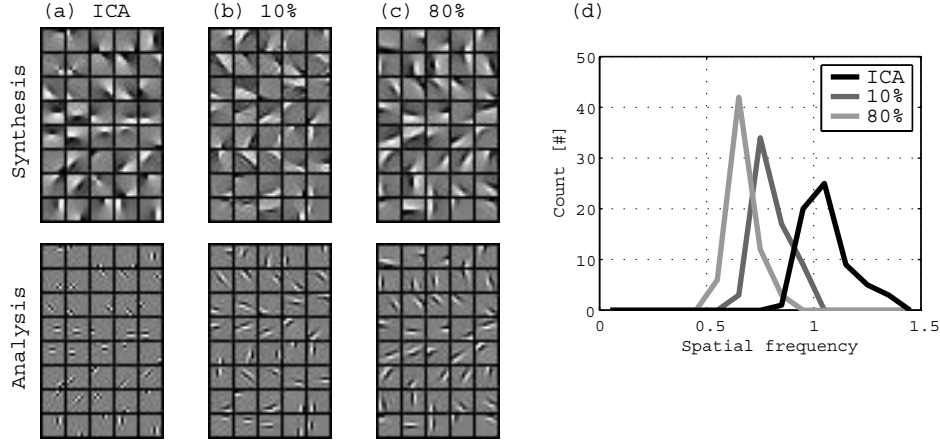

Figure 2: Optimal complete code at different channel noise level. (a-c) Optimized synthesis and analysis vectors. (a) ICA. (b) Proposed model at 10% channel noise level. (c) Proposed model at 80% channel noise level. Here 40 vectors out of 63 are shown. (d) Distribution of spatial-frequency tuning of the analysis vectors in the condition of (a)-(c).

The robustness to channel noise can be explained by the shift of the representation towards lower spatial-frequencies. We analyzed the reconstruction error by projecting it to the principal axes of the data. Figure 3a shows the error spectrum of the code for 80% channel noise, along with the data spectrum (the percentage of the data variance along the principal axes). Note that the data variance of natural images is mostly explained by the first principal components, which correspond to lower spatial-frequencies. In the proposed model, the ratio of the error to the data variance is relatively small around the first principal components. It can be seen much clearer in Figure 3b, where the reconstruction percentage at each principal component is replotted. The reconstruction is more precise for more significant principal components (i.e., smaller index), and it drops down to zero for minor components. For comparison, we analyzed the error for ICA code, where the synthesis and analysis vectors are optimized without channel noise and its robustness to channel noise is tested with 80% channel noise level. As shown in Figure 3, ICA reconstructs every component equally irrespective of their very different data variance[3], therefore the percentage of reconstruction is flat. The proposed model can be robust to channel noise by primarily representing the principal components.

Note that such a biased reconstruction depends on the channel noise level. In Figure 3b we also shows the reconstruction spectrum with 10% channel noise using the code for 10% channel noise level. Compared to the 80% case, the model comes to reconstruct the data at relatively minor components as well. It means that the model can represent finer information if the information capacity of a single unit is large enough. Such a shift of representation is also demonstrated with the toy probems in Figure 1a-c.

### 3.2 Optimal code at different degree of overcompleteness

Now we examine how the optimal representation changes with the different number of available units. We fixed the channel noise level at 80% and vary the degree of overcompleteness as 1x, 2x, 4x, and 8x. Learned vectors for 8x are shown in Figure 4a, and those for

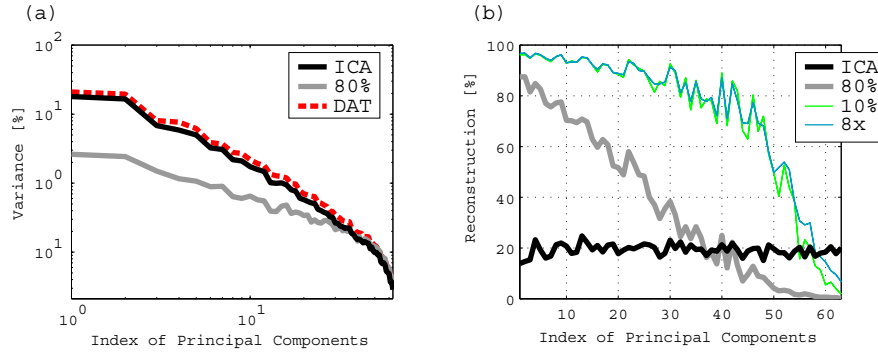

Figure 3: Error analysis. (a) Power spectrum of the data ('DAT') and the reconstruction error with 80% channel noise. '80%' is the error of the 1x code for 80% channel noise level. 'ICA' is the error of the ICA code. (b) Percentage of reconstruction at each principal component. In addition to the conditions in (a), we also show the following (see text). '10%': 1x code for 10% channel noise level. The error is measured with 10% channel noise. '8x': 8x code for 80% channel noise level. The error is measured with 80% channel noise.

1x are in Figure 2c. Compared to the 1x case, where the synthesis and analysis vectors look uniform in shape, the 8x code shows more diversity. To be precise, as illustrated in Figure 4b, the spatial-frequency tuning of the analysis vectors becomes more broadly distributed and cover a larger region as the degree of overcompleteness increases. Physiological data at the central fovea shows that the spatial-frequency tuning of V1 simple cells spans three [11] or two [12] octaves. Models for efficient coding, especially ICA which provides the most efficient code, do not reproduce such a multi-scale representation; instead, the resulting analysis vectors tune only to the highest spatial-frequency (Figure 2a; [3, 9, 10, 7]). It is important that the proposed model generates a broader tuning distribution under the presence of the channel noise and with the high degree of overcompleteness.

An important property of the proposed model is that the reconstruction error decreases as the degree of overcompleteness increases. The resulting error is 23.8, 15.5, 9.7, and 6.2% for 1x, 2x, 4x, and 8x code. The noise analysis shows that the model comes to represent minor components as the degree of overcompleteness increases (Figure 3b). There is an interesting similarity between the error spectra of 8x code for 80% channel noise and 1x code for 10% channel noise. It is suggested that the population of units can represent the same amount and the same kind of information using N times larger number of units if the information capacity of a single unit is decreased with N times larger channel noise level.

## 4 Discussion

A multi-scale representation is known to provide an approximately efficient representation, although it is not optimal as there are known statistical dependencies between scales [13]. We conjecture these residual dependences may be one reason why previous efficient coding models could not yield a broad multi-scale representation. In contrast, the proposed model can introduce useful redundancies in the representation, which is consistent with the emergence of a multi-scale representation. Although it can generate a broader distribution of the spatial-frequency tuning, in these experiments, it covers only about one octave, not two or three octaves as in the physiological data [11, 12]. This issue still remains to be explained.

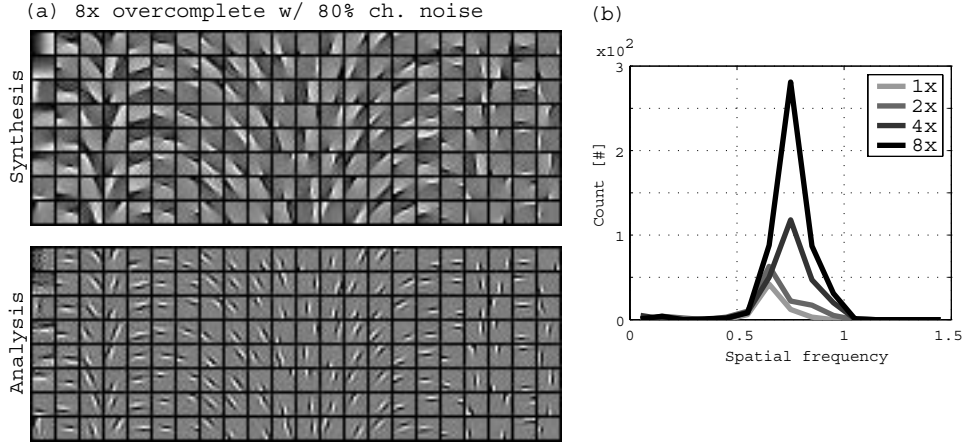

Figure 4: Optimal overcomplete code. (a) Optimized 8x overcomplete code for 80% channel noise level. Here only 176 out of 504 functions are shown. The functions are sorted according to the spatial-frequency tuning of the analysis vectors. (b) Distribution of spatial-frequency tuning of the analysis vectors at different degree of overcompleteness.

Another important characteristic of simple cells is the fact that the more numerous cells are tuned to the lower spatial-frequencies [11, 12]. An explanation of it is that the high data-variance components should be highly oversampled so that the reconstruction erorr is minimized given the limited precision of a single unit [12]. As we described earlier, such a biased representation for the high variance components is observed in our model (Figure 3b). However, the distribution of the spatial-frequency tuning of the analysis vectors does not correspond to this trend; instead, it is bell-shaped (Figure 4b). This apparent inconsistency might be resolved by considering the synthesis vectors, because the reconstruction error is determined by both synthesis and analysis vectors.

A related work is the Atick & Redlich's model for retinal ganglion cells [14]. It also utilizes redundancy in the representation but to compensate for *sensory* noise rather than *channel* noise; therefore, the two models explain different phenomena. Another related work is Olshausen & Field's sparse coding model for simple cells [2], but this again looks at the effects of sensory noise (note that if the sensory noise is neglegible this algorithm does not learn a sparse representation, while the proposed model is appropriate for this condition; of course such a condition might be unrealistic). Now, given a photopic environment where the sensory noise can reasonably be regarded to be small [14], it should rather be important to examine how the constraint of noisy, limited information capacity units changes the representation. It is reported that the information capacity is significantly decreased from photoreceptors to spiking neurons [15], which supports our approach. In spite of its significance, to our knowledge the influence of channel noise on the representation had not been explored.

## 5  Conclusion

We propose a model that both utilizes redundancy in the representation in order to compensate for the limited precision of a single unit and reduces unnecessary redundancy in order to yield an efficient code. The noisy, overcomplete code for natural images generates a distributed spatial-frequency tuning in addition to the Gabor-like analysis vectors, showing a closer agreement with the physiological data than the previous efficient coding models.

The information capacity of a representation may be constrained either by the intrinsic noise in a single unit or by the number of units. In either case, the proposed model can adapt the parameters to primarily represent the high-variance, coarse information, yielding a robust representation to channel noise. As the limitation is relaxed by decreasing the channel noise level or by increasing the number of units, the model comes to represent low-variance, fine information.

## Footnotes

[1]In the noiseless and complete case, they are equivalent to the basis functions [2, 3]. In our setting, however, they are in general no longer basis functions. To make this clear, we call $\mathbf{A}$ and $\mathbf{W}$ as synthesis and analysis vectors.

[2]Since the mean squared error is expressed as $\langle \|\mathbf{e}\|^2 \rangle = \sigma_n^2 \cdot \mathrm{Tr}(\mathbf{AA}^T) = \sigma_n^2 \cdot \mathrm{Tr}(\langle \mathbf{xx}^T \rangle) = \sigma_n^2 \cdot$ (data variance), where $\mathbf{W}$ is whitening filters, $\mathbf{A}(\equiv \mathbf{W}^{-1})$ is their corresponding basis functions. We used eq. (6) and $\langle \mathbf{xx}^T \rangle = \mathbf{AW} \langle \mathbf{xx}^T \rangle \mathbf{W}^T \mathbf{A}^T = \mathbf{AA}^T$.

[3]Since the error spectrum for a whitened representation is expressed as $(\mathbf{E}^t\mathbf{e})^2 = \sigma_n^2 \cdot \mathrm{Diag}(\mathbf{E}^T\langle\mathbf{x}\mathbf{x}^T\rangle\mathbf{E}) = \sigma_n^2 \cdot \mathrm{Diag}(\mathbf{D}) = \sigma_n^2 \cdot (\text{data spectrum})$, where $\mathbf{E}\mathbf{D}\mathbf{E}^T = \langle\mathbf{x}\mathbf{x}^T\rangle$ is the eigen value decomposition of the data covariance matrix.

## References

[1] H. B. Barlow. Possible principles underlying the transformation of sensory messages. In W. A. Rosenblith, editor, *Sensory communication*, pages 217–234. MIT Press, MA, 1961.

[2] B. A. Olshausen and D. J. Field. Sparse coding with an overcomplete basis set: A strategy employed by V1? *Vision Research*, 37:3311–3325, 1997.

[3] A. J. Bell and T. J. Sejnowski. The independent components of natural scenes are edge filters. *Vision Research*, 37:3327–3338, 1997.

[4] H. B. Barlow. Redundancy reduction revisited. *Network: Comput. Neural Syst.*, 12:241–253, 2001.

[5] A. Borst and F. E. Theunissen. Information theory and neural coding. *Nature Neuroscience*, 2:947–957, 1999.

[6] M. S. Lewicki and B. A. Olshausen. Probabilistic framework for the adaptation and comparison of image codes. *J. Opt. Soc. Am. A*, 16:1587–1601, 1999.

[7] E. Doi, T. Inui, T.-W. Lee, T. Wachtler, and T. J. Sejnowski. Spatiochromatic receptive field properties derived from information-theoretic analyses of cone mosaic responses to natural scenes. *Neural Computation*, 15:397–417, 2003.

[8] A. Hyvarinen, J. Karhunen, and E. Oja. *Independent Component Analysis*. John Wiley & Sons, NY, 2001.

[9] J. H. van Hateren and A. van der Schaaf. Independent component filters of natural images compared with simple cells in primary visual cortex. *Proc. R. Soc. Lond. B*, 265:359–366, 1998.

[10] D. L. Ringach. Spatial structure and symmetry of simple-cell receptive fields in macaque primary visual cortex. *Journal of Neurophysiology*, 88:455–463, 2002.

[11] R. L. De Valois, D. G. Albrecht, and L. G. Thorell. Spatial frequency selectivity of cells in macaque visual cortex. *Vision Research*, 22(545-559), 1982.

[12] C. H. Anderson and G. C. DeAngelis. Population codes and signal to noise ratios in primary visual cortex. In *Society for Neuroscience Abstract*, page 822.3, 2004.

[13] E. P. Simoncelli. Modeling the joint statistics of images in the wavelet domain. In *Proc. SPIE 44th Annual Meeting*, pages 188–195, Denver, Colorado, 1999.

[14] J. J. Atick and A. N. Redlich. What does the retina know about natural scenes? *Neural Computation*, 4:196–210, 1992.

[15] S. B. Laughlin and R. R. de Ruyter van Steveninck. The rate of information transfer at graded-potential synapses. *Nature*, 379(15):642–645, 1996.
